# Direct Classification with Indirect Data

**Timothy X Brown**
Interdisciplinary Telecommunications Program
Dept. of Electrical and Computer Engineering
University of Colorado, Boulder, 80309-0530
timxb@colorado.edu

## Abstract

We classify an input space according to the outputs of a real-valued function. The function is not given, but rather examples of the function. We contribute a consistent classifier that avoids the unnecessary complexity of estimating the function.

## 1 Introduction

In this paper, we consider a learning problem that combines elements of regression and classification. Suppose there exists an unknown real-valued property of the feature space, $p(\phi)$, that maps from the feature space, $\phi \in R^n$, to $R$. The property function and a *positive set* $A \subset R$, define the desired classifier as follows:

$$C^*(\phi) = \begin{cases} +1 & \text{if } p(\phi) \in A \\ -1 & \text{otherwise} \end{cases} \qquad (1)$$

Though $p(\phi)$ is unknown, measurements, $\mu$, associated with $p(\phi)$ at different features, $\phi$, are available in a data set $X = \{(\phi_i, \mu_i)\}$ of size $|X| = N$. Each sample is i.i.d. with unknown distribution $f(\phi, \mu)$. This data is indirect in that $\mu$ may be an input to a sufficient statistic for estimating $p(\phi)$ but in itself does not directly indicate $C^*(\phi)$ in (1). Figure 1 gives a schematic of the problem.

Let $C_X(\phi)$ be a decision function mapping from $R^n$ to $\{-1, 1\}$ that is estimated from the data $X$. The estimator, $C_X(\phi)$ is *consistent* if,

$$\lim_{|X| \to \infty} P\{C_X(\phi) \neq C^*(\phi)\} = 0. \qquad (2)$$

where the probabilities are taken over the distribution $f$.

This problem arises in controlling data networks that provide quality of service guarantees such as a maximum packet loss rate [1]–[8]. A data network occasionally drops packets due to congestion. The loss rate depends on the traffic carried by the network (i.e. the network state). The network can not measure the loss rate directly, but can collect data on the observed number of packets sent and lost at different network states. Thus, the feature space, $\phi$, is the network state; the property function, $p(\phi)$, is the underlying loss rate; the measurements, $\mu$, are the observed

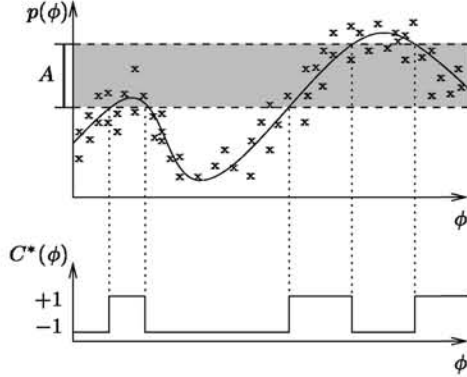

Figure 1: The classification problem. The classifier indicates whether an unknown function, $p(\phi)$, is within a set of interest, $A$. The learner is only given the data "x".

packet losses; the positive set, $A$, is the set of loss rates less than the maximum loss-rate; and the distribution, $f$, follows from the arrival and departures processes of the traffic sources. In words, this application seeks a consistent estimator of when the network can and can not meet the packet loss rate guarantee based on observations of the network losses. Over time, the network can automatically collect a large set of observations so that consistency guarantees the classifier will be accurate.

Previous authors have approached this problem. In [6, 7], the authors estimate the property function from $X$ as, $\hat{p}(\phi)$ and then classify via

$$C(\phi) = \begin{cases} +1 & \text{if } \hat{p}(\phi) \in A \\ -1 & \text{otherwise .} \end{cases} \qquad (3)$$

The approach suffers two related disadvantages. First, an accurate estimate of the property function may require many more parameters than the corresponding classifier in which only the decision boundary is important. Second, the regression requires many samples over the entire range of $\phi$ to be accurate, while the fewer parameters in the classifier may require fewer samples for the same accuracy.

A second approach, used in [4, 5, 8], makes a single sample estimate, $\hat{p}(\phi_i)$ from $\mu_i$ and estimates the desired output class as

$$o_i = \begin{cases} +1 & \text{if } \hat{p}(\phi_i) \in A \\ -1 & \text{otherwise .} \end{cases} \qquad (4)$$

This forms a training set $Y = \{\phi_i, o_i\}$ for standard classification. This was shown to lead to an inconsistent estimator in the data network application in [1].

This paper builds on earlier results by the author specific to the packet network problem [1, 2, 3] and defines a general framework for mapping the indirect data into a standard supervised learning task. It defines conditions on the training set, classifier, and learning objective to yield consistency. The paper defines specific methods based on these results and provides examples of their application.

## 2    Estimator at a Single Feature

In this section, we consider a single feature vector $\phi$ and imagine that we can collect as much monitoring data as we like at $\phi$. We show that a consistent estimator of the property function, $p(\phi)$, yields a consistent estimator of the optimal classification, $C^*(\phi)$, without directly estimating the property function. These results are a basis for the next section where we develop a consistent classifier over the entire feature space even if every $\phi_i$ in the data set is distinct.

Given the data set $X = \{\phi, \mu_i\}$, we hypothesize that there is a mapping from data set to training set $Y = \{\phi, w_i, o_i\}$ such that $|X| = |Y|$ and

$$C_X(\phi) = \text{sign}(\sum_{i=1}^{|X|} w_i o_i) \tag{5}$$

is consistent in the sense of (2). The $w_i$ and $o_i$ are both functions of $\mu_i$, but for simplicity we will not explicitly denote this.

Do any mappings from $X$ to $Y$ yield consistent estimators of the form (5)? We consider only thresholds on $p(\phi)$. That is, sets $A$ in the form $A = [-\infty, \tau)$ (or similarly $A = (\tau, \infty]$) for some threshold $\tau$. Since most practical sets can be formed from finite union, intersection, and complements of sets in this form, this is sufficient.

Consider an estimator $\hat{p}_X$ that has the form

$$\hat{p}_X = \frac{\sum_i \beta(\mu_i)}{\sum_i \alpha(\mu_i)} \tag{6}$$

for some functions $\alpha > 0$, and estimator $\beta$. Suppose that $\hat{p}_X$ is a consistent estimator of $p(\phi)$, i.e. for every $\epsilon > 0$:

$$\lim_{|X| \to \infty} P\{|\hat{p}_X - p(\phi)| > \epsilon\} = 0. \tag{7}$$

For threshold sets such as $A = [-\infty, \tau)$, we can use (6) to construct the classifier:

$$C_X(\phi) = \text{sign}(\tau - \hat{p}_X(\phi)) = \text{sign}\left(\sum_{i=1}^{|X|}(\alpha(\mu_i)\tau - \beta(\mu_i))\right) = \text{sign}(\sum_{i=1}^{|X|} w_i o_i) \tag{8}$$

where

$$w_i = |\alpha(\mu_i)\tau - \beta(\mu_i)| \tag{9}$$
$$o_i = \text{sign}(\alpha(\mu_i)\tau - \beta(\mu_i)) \tag{10}$$

If $|\tau - p(\phi)| = \epsilon$ then the above estimator can be incorrect only if $|\hat{p}_X - p(\phi)| > \epsilon$. The consistency in (7) guarantees that (8)–(10) is consistent if $\epsilon > 0$.

The simplest example of (6) is when $\mu_i$ is a noisy unbiased sample of $p(\phi_i)$. The natural estimator is just the average of all the $\mu_i$, i.e. $\alpha(\mu_i) = 1$ and $\beta(\mu_i) = \mu_i$. In this case, $w_i = |\tau - \mu_i|$ and $o_i = \text{sign}(\tau - \mu_i)$. A less trivial example will be given later in the application section of the paper.

We now describe a range of objective functions for evaluating a classifier $C(\phi; \theta)$ parameterized by $\theta$ and show a correspondence between the objective minimum and (5). Consider the class of weighted $L$-norm objective functions ($L > 0$):

$$J(X, \theta) = \left(\sum_{i=1}^{|X|} w_i |C(\phi; \theta) - o_i|^L\right)^{1/L} \tag{11}$$

Let the $\theta$ that minimizes this be denoted $\theta(X)$. Let

$$C_X(\phi) = C(\phi; \theta(X)) \tag{12}$$

For a single $\phi$, $C(\phi; \theta)$ is a constant $+1$ or $-1$. We can simply try each value and see which is the minimum to find $C_X(\phi)$. This is carried out in [3] where we show:

**Theorem 1** *When $C(\phi; \theta)$ is a constant over $X$ then the $C_X(\phi)$ defined by (11) and (12) is equal to the $C_X(\phi)$ defined by (5).*

The definition in (5) is independent of $L$. So, we can choose any $L$-norm as convenient without changing the solution. This follows since (11) is essentially a weighted count of the errors. The $L$-norm has no significant effect.

This section has shown how regression estimators such as (6) can be mapped via (9) and (10) and the objective (11) to a consistent classifier at a single feature. The next section considers general classifiers.

## 3    Classification over All Features

This section addresses the question of whether there exist any general approach to supervised learning that leads to a consistent estimator across the feature space. Several considerations are important. First, not all feature vectors, $\phi$, are relevant. Some $\phi$ may have zero probability associated with them from the distribution $f(\phi, \mu)$. Such $\phi$ we denote as *unsupported*. The optimal and learned classifier can differ on unsupported feature vectors without affecting consistency. Second, the classifier function $C(\phi, \theta)$ may not be able to represent the consistent estimator. For instance, a linear classifier may never yield a consistent estimator if the optimal classifier, $C^*(\phi)$, decision boundary is non-linear. Classifier functions that can represent the optimal classifier for all supported feature vectors we denote as *representative*. Third, the optimal classifier is discontinuous at the decision boundary. A classifier that considers any small region around a feature on the decision boundary will have both positive and negative samples. In general, the resulting classifier could be $+1$ or $-1$ without regard to the underlying optimal classifier at these points and consistency can not be guaranteed. These considerations are made more precise in Appendix A. Taking these considerations into account and defining $w_i$ and $o_i$ as in (9) and (10) we get the following theorem:

**Theorem 2** *If the classifier (5) is a consistent estimator for every supported non-boundary $\phi$, and $C(\phi; \theta)$ is representative, then the $\theta(X))$ that minimizes (11) yields a consistent classifier over all supported $\phi$ not on the decision boundary.*

Theorem 2 tells us that we can get consistency across the feature space. This result is proved in Appendix A.

## 4    Application

This section provides an application of the results to better illustrate the methodology. For brevity, we include only a simple stylized example (see [3] for a more realistic application). We describe first how the data is created, then the form of the consistent estimator, and then the actual application of the learning method.

The feature space is one dimensional with $\phi$ uniformly distributed in $(3, 9)$. The underlying property function is $p(\phi) = 10^{-\phi}$. The measurement data is generated as follows. For a given $\phi_i$, $s_i$ is the number of successes in $T_i = 10^5$ Bernoulli trials with success probability $p(\phi_i)$. The monitoring data is thus, $\mu_i = (s_i, T_i)$. The positive set is $A = (0, \tau)$ with $\tau = 10^{-6}$, and $|X| = 1000$ samples.

As described in Section 1, this kind of data appears in packet networks where the underlying packet loss rate is unknown and the only monitoring data is the number of packets dropped out of $T_i$ trials. The Bernoulli trial successes correspond to dropped packets. The feature vector represents data collected concurrently that indicates the network state. Thus the classifier can decide when the network will and will not meet a packet loss rate guarantee.

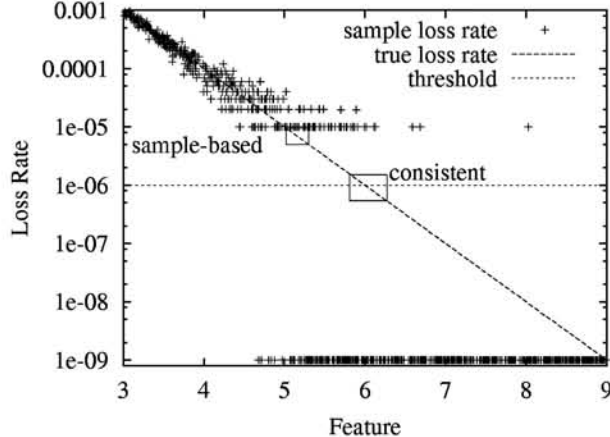

Figure 2: Monitoring data, true property function, and learned classifiers in the loss-rate classification application. The monitoring data is shown as sample loss rate as a function of feature vector. Sample loss-rates of zero are arbitrarily set to $10^{-9}$ for display purposes. The true loss rate is the underlying property function. The consistent and sample-based classifier results are shown as a a range of thresholds on the feature. An $x$ and $y$ error range is plotted as a box. The $x$ error range is the 10th and 90th percentile of 1000 experiments. This is mapped via the underlying property function to a $y$-error range. The consistent classifier finds thresholds around the true value. The sample-based is off by a factor of 7.

Figure 2 shows a sample of data. A consistent estimator in the form of (6) is:

$$\hat{p}_X = \frac{\sum_i s_i}{\sum_i T_i}. \tag{13}$$

Defining $w_i$ and $o_i$ as in (9) and (10) the classifier for our data set is the threshold on the feature space that minimizes (11). This classifier is representative since $p(\phi)$ is monotonic.

The results are shown in Figure 2 and labeled "consistent". This paper's methods find a threshold on the feature that closely corresponds to the $\tau = 10^{-6}$ threshold. As a comparison we also include a classifier that uses $w_i = 1$ for all $i$ and sets $o_i$ to the single-sample estimate, $\hat{p}(\phi_i) = s_i/T_i$, as in (4). The results are labeled "sample-based". This method misses the desired threshold by a factor of 7.

This application shows the features of the paper's methods. The classifier is a simple threshold with one parameter. Estimating $p(\phi)$ to derive a classifier required 10's of parameters in [6, 7]. The results are consistent unlike the approaches in [4, 5, 8].

## 5  Conclusion

This paper has shown that using indirect data we can define a classifier that directly uses the data without any intermediate estimate of the underlying property function. The classifier is consistent and yields a simpler learning problem. The approach was demonstrated on a problem from telecommunications. Practical details such as choosing the form of the parametric classifier, $C(\phi; \theta)$, or how to find the global minimum of the objective function (11) are outside the scope of this paper.

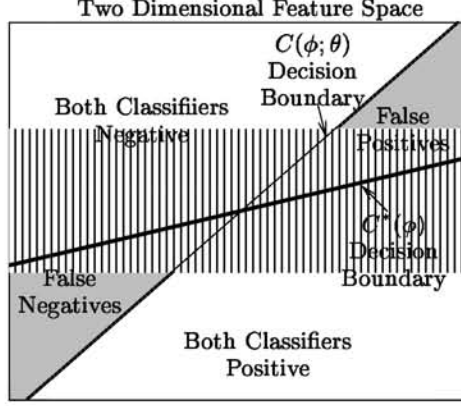

Figure 3: A classifier $C(\phi;\theta)$ and the optimal classifier $C^*(\phi)$ create four different sets in feature space: where they agree and are positive; where they agree and are negative; where they disagree and $C^*(\phi) = +1$ (false negatives); and where they disagree and $C^*(\phi) = -1$ (false positives).

## A  Appendix: Consistency of Supervised Learning

This appendix proves certain natural conditions on a supervised learner lead to a consistent classifier (Theorem 2). First we need to formally define several concepts.

Since the feature space is real, it is a metric space with measure $m$.

A feature vector $\phi$ is *supported* by the distribution $f$ if every neighborhood around $\phi$ has positive probability.

A feature vector $\phi$ is on the *decision boundary* if in every neighborhood around $\phi$ there exists supported $\phi'$, $\phi''$ such that $C^*(\phi') \neq C^*(\phi'')$.

A classifier function, $C(\phi;\theta)$ is *representative* if there exists a $\theta^*$ such that $C(\phi;\theta^*) = C^*(\phi)$ for all supported, non-boundary $\phi$.

Parameters $\theta$ and $\theta'$ are *equivalent* if for all supported, non-boundary $\phi$; $C(\phi;\theta) = C(\phi;\theta')$.

Given a $\theta$, it is either equivalent to $\theta^*$ or there are supported, non-boundary $\phi$ where $C(\phi;\theta)$ is not equal to the optimal classifier as in Figure 3. We will show that for any $\theta$ not equivalent to $\theta^*$,

$$\lim_{|X| \to \infty} P\{J(X,\theta) \leq J(X,\theta^*)\} = 0 \tag{14}$$

In other words, such a $\theta$ can not be the minimum of the objective in (11) and so only a $\theta$ equivalent to $\theta^*$ is a possible minimum.

To prove Theorem 2, we need to introduce a further condition. An estimator of the form (5) has uniformly bounded variance if $\mathrm{Var}(w_i) < B$ for some fixed $B < \infty$ for all $\phi$.

Let $E[w(\phi)o(\phi)] = e(\phi)$ be the expected weighted desired output for independent samples at $\phi$ where the expectation is from $f(\mu|\phi)$. To start, we note that if (5) is consistent, then:

$$\mathrm{sign}(e(\phi)) = C^*(\phi) \tag{15}$$

for all non-boundary states. Looking at Figure 3, let us focus on the false negative set minus the optimal decision boundary, call this $\Phi$. From (15), $e(\phi)$ is positive for every $\phi \in \Phi$. Let $x$ be the probability measure of $\Phi$. Define the set

$$\Phi_\epsilon = \{\phi | \phi \in \Phi \text{ and } e(\phi) \geq \epsilon\}.$$

Let $x_\epsilon$ be the probability measure of $\Phi_\epsilon$. Choose $\epsilon > 0$ so that $x_\epsilon > 0$.

The proof is straight forward from here and we omit some details. With $\theta$, $C(\phi; \theta) = -1$ for all $\phi \in \Phi$. With $\theta^*$, $C(\phi; \theta^*) = +1$ for all $\phi \in \Phi$. Since the minimum of a constant objective function satisfies (5), we would incorrectly choose $\theta$ if

$$\lim_{|X| \to \infty} \sum_{i=1}^{|X|} w_i o_i < 0$$

For the false negatives the expected number of examples in $\Phi$ and $\Phi_\epsilon$ is $x|X|$ and $x_\epsilon |X|$. By the definition of $\Phi_\epsilon$ and the bounded variance of the weight, we get that

$$E[\sum_{i=1}^{|X|} w_i o_i] \geq \epsilon x_\epsilon |X| \tag{16}$$

$$\text{Var}[\sum_{i=1}^{|X|} w_i o_i] < Bx|X|. \tag{17}$$

Since the expected value grows linearly with the sample size and the standard deviation with the square root of the sample size, as $|X| \to \infty$ the weighted sum will with probability one be positive. Thus, as the sample size grows, $+1$ will minimize the objective function for the set of false negative samples and the decision boundary from $\theta^*$ will minimize the objective.

The same argument applied to the false positives shows that $\theta^*$ will minimize the false positives with probability one. Thus $\theta^*$ will be chosen with probability one and the theorem is shown.

## Acknowledgments

This work was supported by NSF CAREER Award NCR-9624791.

## References

[1] Brown, T.X (1995) Classifying loss rates with small samples, *Proc. Inter. Workshop on Appl. of NN to Telecom* (pp. 153–161). Hillsdale, NJ: Erlbaum.

[2] Brown, T.X (1997) Adaptive access control applied to ethernet data, *Advances in Neural Information Processing Systems, 9* (pp. 932–938). MIT Press.

[3] Brown, T. X (1999) Classifying loss rates in broadband networks, *INFOCOMM '99* (v. 1, pp. 361–370). Piscataway, NJ: IEEE.

[4] Estrella, A.D., et al. (1994). New training pattern selection method for ATM call admission neural control, *Elec. Let.*, v. 30, n. 7, pp. 577–579.

[5] Hiramatsu, A. (1990). ATM communications network control by neural networks, *IEEE T. on Neural Networks*, v. 1, n. 1, pp. 122–130.

[6] Hiramatsu, A. (1995). Training techniques for neural network applications in ATM, *IEEE Comm. Mag.*, October, pp. 58–67.

[7] Tong, H., Brown, T. X (1998). Estimating Loss Rates in an Integrated Services Network by Neural Networks, *Proc. of Global Telecommunications Conference (GLOBECOM 98)* (v. 1, pp. 19–24) Piscataway, NJ: IEEE.

[8] Tran-Gia, P., Gropp, O. (1992). Performance of a neural net used as admission controller in ATM systems, *Proc. GLOBECOM 92* (pp. 1303–1309). Piscataway, NJ: IEEE.
